# Approximate Expectation Maximization

**Tom Heskes, Onno Zoeter, and Wim Wiegerinck**
SNN, University of Nijmegen
Geert Grooteplein 21, 6525 EZ, Nijmegen, The Netherlands

## Abstract

We discuss the integration of the expectation-maximization (EM) algorithm for maximum likelihood learning of Bayesian networks with belief propagation algorithms for approximate inference. Specifically we propose to combine the outer-loop step of convergent belief propagation algorithms with the M-step of the EM algorithm. This then yields an approximate EM algorithm that is essentially still double loop, with the important advantage of an inner loop that is guaranteed to converge. Simulations illustrate the merits of such an approach.

## 1   Introduction

The EM (expectation-maximization) algorithm [1, 2] is a popular method for maximum likelihood learning in probabilistic models with hidden variables. The E-step boils down to computing probabilities of the hidden variables given the observed variables (evidence) and current set of parameters. The M-step then, given these probabilities, yields a new set of parameters guaranteed to increase the likelihood. In Bayesian networks, that will be the focus of this article, the M-step is usually relatively straightforward. A complication may arise in the E-step, when computing the probability of the hidden variables given the evidence becomes intractable.

An often used approach is to replace the exact yet intractable inference in the E-step with approximate inference, either through sampling or using a deterministic variational method. The use of a "mean-field" variational method in this context leads to an algorithm known as variational EM and can be given the interpretation of minimizing a free energy with respect to both a tractable approximate distribution (approximate E-step) and the parameters (M-step) [2].

Loopy belief propagation [3] and variants thereof, such as generalized belief propagation [4] and expectation propagation [5], have become popular alternatives to the "mean-field" variational approaches, often yielding somewhat better approximations. And indeed, they can and have been applied for approximate inference in the E-step of the EM algorithm (see e.g. [6, 7]). A possible worry, however, is that standard application of these belief propagation algorithms does not always lead to convergence. So-called double-loop algorithms with convergence guarantees have been derived, such as CCCP [8] and UPS [9], but they tend to be an order of magnitude slower than standard belief propagation.

The goal of this article is to integrate expectation-maximization with belief propagation. As for variational EM, this integration relies on the free-energy interpretation

of EM that is reviewed in Section 2. In Section 3 we describe how the exact free energy can be approximated with a Kikuchi free energy and how this leads to an approximate EM algorithm. Section 4 contains our main result: integrating the outer-loop of a convergent double-loop algorithm with the M-step, we are left with an overall double-loop algorithm, where the inner loop is now a convex constrained optimization problem with a unique solution. The methods are illustrated in Section 5; implications and extensions are discussed in Section 6.

## 2   The free energy interpretation of EM

We consider probabilistic models $P(x;\theta)$, with $\theta$ the model parameters to be learned and $x$ the variables in the model. We subdivide the variables into hidden variables $h$ and observed, evidenced variables $e$. For ease of notation, we consider just a single set of observed variables $e$ (in fact, if we have $N$ sets of observed variables, we can simply copy our probability model $N$ times and view this as our single probability model with "shared" parameters $\theta$). In maximum likelihood learning, the goal is to find the parameters $\theta$ that maximize the likelihood $P(e;\theta)$ or, equivalently, that minimize minus the loglikelihood

$$L(\theta) = -\log P(e;\theta) = -\log \left[ \sum_h P(e,h;\theta) \right] .$$

The EM algorithm can be understood from the observation, made in [2], that

$$L(\theta) = \min_{Q \in \mathcal{P}} F(Q,\theta) ,$$

with $\mathcal{P}$ the set of all probability distributions defined on $h$ and $F(Q,\theta)$ the so-called free energy

$$F(Q,\theta) = L(\theta) + \sum_h Q(h) \log \left[ \frac{Q(h)}{P(h|e;\theta)} \right] = E(Q,\theta) - S(Q) , \qquad (1)$$

with the "energy"

$$E(Q,\theta) = -\sum_h Q(h) \log P(e,h;\theta) ,$$

and the "entropy"

$$S(Q) = -\sum_h Q(h) \log Q(h) .$$

The EM algorithm now boils down to alternate minimization with respect to $Q$ and $\theta$:

$$
\begin{aligned}
&\textit{E-step:} \quad \text{fix } \theta \text{ and solve} \quad Q = \operatorname*{argmin}_{Q' \in \mathcal{P}} F(Q',\theta) \\
&\textit{M-step:} \quad \text{fix } Q \text{ and solve} \quad \theta = \operatorname*{argmin}_{\theta'} F(Q,\theta') = \operatorname*{argmin}_{\theta'} E(Q,\theta')
\end{aligned}
\qquad (2)
$$

The advantage of the M-step over direct minimization of $-\log P(e;\theta)$ is that the summation over $h$ is now outside the logarithm, which in many cases implies that the minimum with respect to $\theta$ can be computed explicitly. The main inference problem is then in the E-step. Its solution follows directly from (1):

$$Q(h) = P(h|e;\theta) = \frac{P(h,e;\theta)}{\sum_{h'} P(h',e;\theta)} , \qquad (3)$$

with $\theta$ the current setting of the parameters. However, in complex probability models $P(h|e;\theta)$ can be difficult and even intractable to compute, mainly because

of the normalization in the denominator. For later purposes we note that the EM algorithm can be interpreted as a general "bound optimization algorithm" [10]. In this interpretation the free energy $F(Q, \theta)$ is an upper bound on the function $L(\theta)$ that we try to minimize; the E-step corresponds to a reset of the bound and the M-step to the minimization of the upper bound.

In variational EM [2] one restricts the probability distribution $Q$ to a specific set $\mathcal{P}'$, such that the E-step becomes tractable. Note that this restriction affects both the energy term and the entropy term. By construction the approximate $\min_{Q \in \mathcal{P}'} F(Q, \theta)$ is an upper bound on $L(\theta)$.

## 3  Approximate free energies

In several studies, propagation algorithms like loopy belief propagation [6] and expectation propagation [7] have been applied to find approximate solutions for the E-step. As we will see, the corresponding approximate EM-algorithm can be interpreted as alternate minimization of a Bethe or Kikuchi free energy. For the moment, we will consider the case of loopy and generalized belief propagation applied to probability models with just discrete variables. The generalization to expectation propagation is discussed in Section 6.

The joint probability implied by a Bayesian network can be written in the form

$$P(x; \theta) = \prod_\alpha \Psi_\alpha(x_\alpha; \theta_\alpha) \,,$$

where $\alpha$ denotes a subset of variables and $\Psi_\alpha$ is a potential function. The parameters $\theta_\alpha$ may be shared, i.e., we may have $\theta_\alpha \equiv \theta_{\alpha'}$ for some $\alpha \neq \alpha'$. For a Bayesian network, the energy term simplifies into a sum over local terms:

$$E(Q, \theta) = -\sum_\alpha \sum_{h_\alpha} Q(h_\alpha) \log \Psi_\alpha(h_\alpha, e_\alpha; \theta_\alpha) \,.$$

However, the entropy term is as intractable as the normalization in (3) that we try to prevent. In the Bethe or more generally Kikuchi approximation, this entropy term is approximated through [4]

$$S(Q) = -\sum_h Q(h) \log Q(h) \approx \sum_\alpha S_\alpha(Q) + \sum_\beta c_\beta S_\beta(Q) \equiv \tilde{S}(Q) \,,$$

with

$$S_\alpha(Q) = -\sum_{h_\alpha} Q(h_\alpha) \log Q(h_\alpha) \,,$$

and similarly for $S_\beta(Q)$. The subsets indexed by $\beta$ correspond to intersections between the subsets indexed by $\alpha$, intersections of intersections, and so on. The parameters $c_\beta$ are called Moebius or overcounting numbers. In the above description, the $\alpha$-clusters correspond to the potential subsets, i.e., the clusters in the moralized graph. However, we can also choose them to be larger, e.g., combining several potentials into a single cluster. The Kikuchi/Bethe approximation is exact if the $\alpha$-clusters form a singly-connected structure. That is, exact inference is obtained when the $\alpha$-clusters correspond to cliques in a junction tree. The $\beta$ subsets then play the role of the separators and have overcounting numbers $1 - n_\beta$ with $n_\beta$ the number of neighboring cliques. The larger the clusters, the higher the computational complexity.

There are different kinds of approximations (Bethe, CVM, junction graphs), each corresponding to a somewhat different choice of $\alpha$-clusters, $\beta$-subsets and overcounting numbers (see [4] for an overview). In the following we will refer to all of them

as Kikuchi approximations. The important point is that the approximate entropy is, like the energy, a sum of local terms. Furthermore, the Kikuchi free energy as a function of the probability distribution $Q$ only depends on the marginals $Q(x_\alpha)$ and $Q(x_\beta)$. The minimization of the exact free energy with respect to a probability distribution $Q$ has been turned into the minimization of the Kikuchi free energy $\tilde{F}(Q, \theta) = E(Q, \theta) - \tilde{S}(Q)$ with respect to a set of pseudo-marginals $Q = \{Q_\alpha, Q_\beta\}$. For the approximation to make any sense, these pseudo-marginals have to be properly normalized as well as consistent, which boils down to a set of linear constraints of the form

$$Q_\alpha(x_\beta) = \sum_{x_{\alpha \backslash \beta}} Q_\alpha(x_\alpha) = Q_\beta(x_\beta) \, . \tag{4}$$

The approximate EM algorithm based on the Kikuchi free energy now reads

$$\text{approximate E-step:} \quad \text{fix } \theta \text{ and solve} \quad Q = \underset{Q' \in \tilde{\mathcal{P}}}{\arg\min} \tilde{F}(Q', \theta)$$

$$\text{M-step:} \quad \text{fix } Q \text{ and solve} \quad \theta = \underset{\theta'}{\arg\min} \tilde{F}(Q, \theta') = \underset{\theta'}{\arg\min} E(Q, \theta')$$

$$\tag{5}$$

where $\tilde{\mathcal{P}}$ refers to all sets of consistent and properly normalized pseudo-marginals $\{Q_\alpha, Q_\beta\}$. Because the entropy does not depend on the parameters $\theta$, the M-step of the approximate EM algorithm is completely equivalent to the M-step of the exact EM algorithm. The only difference is that the statistics required for this M-step is computed approximately rather than exactly. In other words, the seemingly naive procedure of using generalized or loopy belief propagation to compute the statistics in the E-step and use it in the M-step, can be interpreted as alternate minimization of the Kikuchi approximation of the exact free energy. That is, algorithm (5) can be interpreted as a bound optimization algorithm for minimizing

$$\tilde{L}(\theta) = \min_{Q \in \tilde{\mathcal{P}}} \tilde{F}(Q, \theta) \, ,$$

which we hope to be a good approximation (not necessarily a bound) of the original $L(\theta)$.

## 4   Constrained optimization

There are two kinds of approaches for finding the minimum of the Kikuchi free energy. The first one is to run loopy or generalized belief propagation, e.g., using Algorithm 1 in the hope that it converges to such a minimum. However, convergence guarantees can only be given in special cases and in practice one does observe convergence problems. In the following we will refer to the use of standard belief propagation in the E-step as the "naive algorithm".

Recently, there have been derived double-loop algorithms that explicitly minimize the Kikuchi free energy [8, 9, 11]. Technically, finding the minimum of the Kikuchi free energy with respect to consistent marginals corresponds to a non-convex constrained optimization problem. The consistency and normalization constraints on the marginals are linear in $Q$ and so is the energy term $E(Q, \theta)$. The non-convexity stems from the entropy terms and specifically those with negative overcounting numbers. Most currently described techniques, such as CCCP [8], UPS [9] and variants thereof, can be understood as general bound optimization algorithms. In CCCP concave terms are bounded with a linear term, yielding a convex bound and thus, in combination with the linear constraints, a convex optimization problem to be solved in the inner loop. In particular we can write

$$\tilde{F}(Q, \theta) = \min_{R \in \tilde{\mathcal{P}}} G(Q, R, \theta) \quad \text{with} \quad G(Q, R, \theta) = \tilde{F}(Q, \theta) + K(Q, R) \, , \tag{6}$$

**Algorithm 1** Generalized belief propagation.

1: **while** ¬converged **do**
2:    **for all** $\beta$ **do**
3:       **for all** $\alpha \supset \beta$ **do**
4: 
$$Q_\alpha(x_\beta) = \sum_{X_{\alpha \backslash \beta}} Q_\alpha(X_\alpha); \qquad \mu_{\alpha \to \beta}(x_\beta) = \frac{Q_\alpha(x_\beta)}{\mu_{\beta \to \alpha}(x_\beta)}$$
5:       **end for**
6: 
$$Q_\beta(x_\beta) \propto \prod_{\alpha \supset \beta} \mu_{\alpha \to \beta}(x_\beta)^{\frac{1}{n_\beta + c_\beta}}$$
7:       **for all** $\alpha \supset \beta$ **do**
8: 
$$\mu_{\beta \to \alpha}(x_\beta) = \frac{Q_\beta(x_\beta)}{\mu_{\alpha \to \beta}(x_\beta)}; \qquad Q_\alpha(X_\alpha) \propto \Psi_\alpha(X_\alpha) \prod_{\beta \subset \alpha} \mu_{\beta \to \alpha}(x_\beta)$$
9:       **end for**
10:    **end for**
11: **end while**

where

$$K(Q, R) \equiv \sum_{\beta; c_\beta < 0} |c_\beta| \sum_{h_\beta} Q_\beta(h_\beta) \log \left[ \frac{Q_\beta(h_\beta)}{R_\beta(h_\beta)} \right] ,$$

is a weighted sum of local Kullback-Leibler divergences. By construction $G(Q, R, \theta)$ is convex in $Q$ - the concave $Q_\beta \log Q_\beta$ terms in $\tilde{F}(Q, \theta)$ cancel with those in $K(Q, R)$ - as well as an upper bound on $\tilde{F}(Q, \theta)$ since $K(Q, R) \geq 0$. The now convex optimization problem in the inner loop can be solved with a message passing algorithm very similar to standard loopy or generalized belief propagation. In fact, we can use Algorithm 1, with $c_\beta = 0$ and after a slight redefinition of the potentials $\Psi_\alpha$ such that they incorporate the linear bound of the concave entropy terms (see [11] for details). The messages in this algorithm are in one-to-one correspondence with the Lagrange multipliers of the concave dual. Most importantly, with the particular scheduling in Algorithm 1, each update is guaranteed to increase the dual and therefore the inner-loop algorithm must converge to its unique solution. The outer loop simply sets $R = Q$ and corresponds to a reset of the bound.

Incorporating this double-loop algorithm into our approximate EM algorithm (5), we obtain

$$
\begin{array}{rll}
\textit{inner-loop E-step:} & \text{fix } \{\theta, R\} \text{ solve} & Q = \underset{Q' \in \tilde{\mathcal{P}}}{\text{argmin}}\, G(Q', R, \theta) \\[1em]
\textit{outer-loop E-step:} & \text{fix } \{Q, \theta\} \text{ solve} & R = \underset{R' \in \tilde{\mathcal{P}}}{\text{argmin}}\, G(Q, R', \theta) = \underset{R' \in \tilde{\mathcal{P}}}{\text{argmin}}\, K(Q, R) \\[1em]
\textit{M-step:} & \text{fix } \{Q, R\} \text{ solve} & \theta = \underset{\theta'}{\text{argmin}}\, G(Q, R, \theta') = \underset{\theta'}{\text{argmin}}\, E(Q, \theta')
\end{array}
$$
$$(7)$$

To distinguish it from the naive algorithm, we will refer to (7) as the "convergent algorithm". The crucial observation is that we can combine the outer-loop E-step with the usual M-step: there is no need to run the double-loop algorithm in the E-step until convergence. This gives us then an overall double-loop rather than triple-loop algorithm. In principle (see however the next section) the algorithmic complexity of the convergent algorithm is the same as that of the naive algorithm.

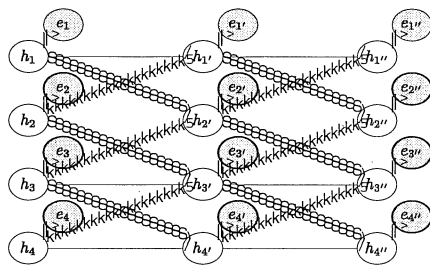

(a) Coupled hidden Markov model.

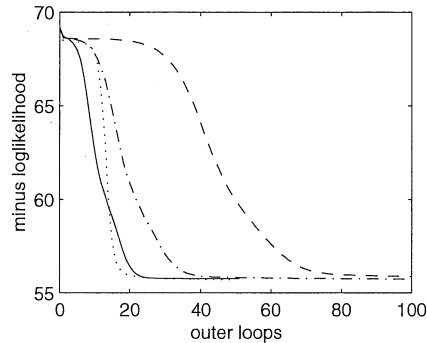

(b) Simulation results.

Figure 1: Learning a coupled hidden Markov model. (a) Architecture for 3 time slice and 4 hidden nodes per time slice. (b) Minus the loglikelihood in the Kikuchi/Bethe approximation as a function of the number of M-steps. Naive algorithm (solid line), convergent algorithm (dashed), convergent algorithm with tighter bound and overrelaxation (dash-dotted), same for a Kikuchi approximation (dotted). See text for details.

## 5  Simulations

For illustration, we compare the naive and convergent approximate EM algorithms for learning in a coupled hidden Markov model. The architecture of coupled hidden Markov models is sketched in Figure 1(a) for $T = 3$ time slices and $M = 4$ hidden-variable nodes per time slice. In our simulations we used $M = 5$ and $T = 20$; all nodes are binary. The parameters to be learned are the observation matrix $p(e_{m,t} = i | h_{m,t} = j)$ and two transition matrices: $p(h_{1,t+1} = i | h_{1,t} = j, h_{2,t} = k) = p(h_{M,t+1} = i | h_{M,t} = j, h_{M-1,t} = k)$ for the outer nodes and $p(h_{m,t+1} = i | h_{m-1,t} = j, h_{m,t} = k, h_{m+1,t} = l)$ for the middle nodes. The prior for the first time slice is fixed and uniform. We randomly generated properly normalized transition and observation matrices and evidence given those matrices. Initial parameters were set to another randomly generated instance. In the inner loop of both the naive and the convergent algorithm, Algorithm 1 was run for 10 iterations.

Loopy belief propagation, which for dynamic Bayesian networks can be interpreted as an iterative version of the Boyen-Koller algorithm [12], converged just fine for the many instances that we have seen. The naive algorithm nicely minimizes the Bethe approximation of minus the loglikelihood $\tilde{L}(\theta)$, as can be seen from the solid line in Figure 1(b). The Bethe approximation is fairly accurate in this model and plots of the exact loglikelihood, both those learned with exact and with approximate EM, are very similar (not shown). The convergent algorithm also works fine, but takes more time to converge (dashed line). This is to be expected: the additional bound implied by the outer-loop E-step makes $G(Q, R, \theta)$ a looser bound of $\tilde{L}(\theta)$ than $\tilde{F}(Q, \theta)$ and the tighter the bound in a bound optimization algorithm, the faster the convergence. Therefore, it makes sense to use tighter convex bounds on $\tilde{F}(Q, \theta)$, for example those derived in [11]. On top of that, we can use overrelaxation, i.e., set $\log Q = \eta \log R + (1 - \eta) \log Q^{\text{old}}$ (up to normalization) with $Q^{\text{old}}$ the previous set of pseudo-marginals. See e.g. [10] for the general idea; here we took $\eta = 1.4$ fixed. Application of these two "tricks" yields the dash-dotted line. It gives an indication of how close one can bring the convergent to the naive algorithm (overrelaxation

applied to the M-step affects both algorithms in the same way and is therefore not considered here). Another option is to repeat the inner and outer E-steps $N$ times, before updating the parameters in the M-step. Plots for $N \geq 3$ are indistinguishable from the solid line for the naive algorithm.

The above shows that the price to be paid for an algorithm that is guaranteed to converge is relatively low. Obviously, the true value of the convergent algorithm becomes clear when the naive algorithm fails. Many instances of non-convergence of loopy and especially generalized belief propagation have been reported (see e.g. [3, 11] and [12] specifically on coupled hidden Markov models). Some but not all of these problems disappear when the updates are damped, which further has the drawback of slowing down convergence as well as requiring additional tuning. In the context of the coupled hidden Markov models we observed serious problems with generalized belief propagation. For example, with $\alpha$-clusters of size 12, consisting of 3 neighboring hidden and evidence nodes in two subsequent time slices, we did not manage to get the naive algorithm to converge properly. The convergent algorithm always converged without any problem, yielding the dotted line in Figure 1(b) for the particular problem instance considered for the Bethe approximation as well. Note that, where the inner loops for the Bethe approximations take about the same amount of time (which makes the number of outer loops roughly proportional to cpu time), an inner loop for the Kikuchi approximation is in this case about two times slower.

# 6  Discussion

The main idea of this article, that there is no need to run a converging double-loop algorithm in an approximate E-step until convergence, only applies to directed probabilistic graphical models like Bayesian networks. In undirected graphical models like Boltzmann machines there is a global normalization constant that typically depends on all parameters $\theta$ and is intractable to compute analytically. For this so-called partition function, the bound used in converging double-loop algorithms works in the opposite direction as the bound implicit in the EM algorithm. The convex bound of [13] does work in the right direction, but cannot (yet) handle missing values. In [14] standard loopy belief propagation is used in the inner loop of iterative proportional fitting (IPF). Also here it is not yet clear how to integrate IPF with convergent belief propagation without ending up with a triple-loop algorithm.

Following the same line of reasoning, expectation maximization can be combined with expectation propagation (EP) [5]. EP can be understood as a generalization of loopy belief propagation. Besides neglecting possible loops in the graphical structure, expectation propagation can also handle projections onto an exponential family of distributions. The approximate free energy for EP is the same Bethe free energy, only the constraints are different. That is, the "strong" marginalization constraints (4) are replaced by the "weak" marginalization constraints that all subsets marginals agree upon their moments. These constraints are still linear in $Q_\alpha$ and $Q_\beta$ and we can make the same decomposition (6) of the Bethe free energy into a convex and a concave term to derive a double-loop algorithm with a convex optimization problem in the inner loop. However, EP can have reasons for non-convergence that are not necessarily resolved with a double-loop version. For example, it can happen that while projecting onto Gaussians negative covariance matrices appear. This problem has, to the best of our knowledge, not yet been solved and is subject to ongoing research.

It has been emphasized before [13] that it makes no sense to learn with approxi-

mate inference and then apply exact inference given the learned parameters. The intuition is that we tune the parameters to the evidence, incorporating the errors that are made while doing approximate inference. In that context it is important that the results of approximate inference are reproducable and the use of convergent algorithms is a relevant step in that direction.

# References

[1] A. Dempster, N. Laird, and D. Rubin. Maximum likelihood from incomplete data via the EM algorithm. *Journal of the Royal Statistical Society B*, 39:1–38, 1977.

[2] R. Neal and G. Hinton. A view of the EM algorithm that justifies incremental, sparse, and other variants. In M. Jordan, editor, *Learning in Graphical Models*, pages 355–368. Kluwer Academic Publishers, Dordrecht, 1998.

[3] K. Murphy, Y. Weiss, and M. Jordan. Loopy belief propagation for approximate inference: An empirical study. In *Proceedings of the Fifteenth Conference on Uncertainty in Articial Intelligence*, pages 467–475, San Francisco, CA, 1999. Morgan Kaufmann.

[4] J. Yedidia, W. Freeman, and Y. Weiss. Constructing free energy approximations and generalized belief propagation algorithms. Technical report, Mitsubishi Electric Research Laboratories, 2002.

[5] T. Minka. Expectation propagation for approximate Bayesian inference. In *Uncertainty in Artificial Intelligence: Proceedings of the Seventeenth Conference (UAI-2001)*, pages 362–369, San Francisco, CA, 2001. Morgan Kaufmann Publishers.

[6] B. Frey and A. Kanna. Accumulator networks: Suitors of local probability propagation. In T. Leen, T. Dietterich, and V. Tresp, editors, *Advances in Neural Information Processing Systems 13*, pages 486–492. MIT Press, 2001.

[7] T. Minka and J. Lafferty. Expectation propagation for the generative aspect model. In *Proceedings of UAI-2002*, pages 352–359, 2002.

[8] A. Yuille. CCCP algorithms to minimize the Bethe and Kikuchi free energies: Convergent alternatives to belief propagation. *Neural Computation*, 14:1691–1722, 2002.

[9] Y. Teh and M. Welling. The unified propagation and scaling algorithm. In *NIPS 14*, 2002.

[10] R. Salakhutdinov and S. Roweis. Adaptive overrelaxed bound optimization methods. In *ICML-2003*, 2003.

[11] T. Heskes, K. Albers, and B. Kappen. Approximate inference and constrained optimization. In *UAI-2003*, 2003.

[12] K. Murphy and Y. Weiss. The factored frontier algorithm for approximate inference in DBNs. In *UAI-2001*, pages 378–385, 2001.

[13] M. Wainwright, T. Jaakkola, and A. Willsky. Tree-reweighted belief propagation algorithms and approximate ML estimation via pseudo-moment matching. In *AISTATS-2003*, 2003.

[14] Y. Teh and M. Welling. On improving the efficiency of the iterative proportional fitting procedure. In *AISTATS-2003*, 2003.
